# Group Sparse Coding

**Samy Bengio**
Google
Mountain View, CA
bengio@google.com

**Fernando Pereira**
Google
Mountain View, CA
pereira@google.com

**Yoram Singer**
Google
Mountain View, CA
singer@google.com

**Dennis Strelow**
Google
Mountain View, CA
strelow@google.com

## Abstract

Bag-of-words document representations are often used in text, image and video processing. While it is relatively easy to determine a suitable word dictionary for text documents, there is no simple mapping from raw images or videos to dictionary terms. The classical approach builds a dictionary using vector quantization over a large set of useful visual descriptors extracted from a training set, and uses a nearest-neighbor algorithm to count the number of occurrences of each dictionary word in documents to be encoded. More robust approaches have been proposed recently that represent each visual descriptor as a sparse weighted combination of dictionary words. While favoring a sparse representation at the level of visual descriptors, those methods however do not ensure that images have sparse representation. In this work, we use mixed-norm regularization to achieve sparsity at the image level as well as a small overall dictionary. This approach can also be used to encourage using the same dictionary words for all the images in a class, providing a discriminative signal in the construction of image representations. Experimental results on a benchmark image classification dataset show that when compact image or dictionary representations are needed for computational efficiency, the proposed approach yields better mean average precision in classification.

## 1   Introduction

Bag-of-words document representations are widely used in text, image, and video processing [14, 1]. Those representations abstract from spatial and temporal order to encode a document as a vector of the numbers of occurrences in the document of descriptors from a suitable dictionary. For text documents, the dictionary might consist of all the words or of all the $n$-grams of a certain minimum frequency in the document collection [1].

For images or videos, however, there is no simple mapping from the raw document to descriptor counts. Instead, visual descriptors must be first extracted and then represented in terms of a carefully constructed dictionary. We will not discuss further here the intricate processes of identifying useful visual descriptors, such as color, texture, angles, and shapes [14], and of measuring them at appropriate document locations, such as on regular grids, on special interest points, or at multiple scales [6].

For dictionary construction, the standard approach in computer vision is to use some unsupervised vector quantization (VQ) technique, often $k$-means clustering [14], to create the dictionary. A new image is then represented by a vector indexed by dictionary elements (codewords), which for element $d$ counts the number of visual descriptors in the image whose closest codeword is $d$. VQ

representations are maximally sparse per descriptor occurrence since they pick a single codeword for each occurrence, but they may not be sparse for the image as a whole; furthermore, such representations are not that robust with respect to descriptor variability.

Sparse representations have obvious computational benefits, by saving both processing time in handling visual descriptors and space in storing encoded images. To alleviate the brittleness of VQ representations, several studies proposed representation schemes where each visual descriptor is encoded as a weighted sum of dictionary elements, where the encoding optimizes a tradeoff between reconstruction error and the $\ell_1$ norm of the reconstruction weights [3, 5, 7, 8, 9, 16]. These techniques promote sparsity in determining a small set of codewords from the dictionary that can be used to efficiently represent each visual descriptor of each image [13].

However, those approaches consider each visual descriptor in the image as a separate coding problem and do not take into account the fact that descriptor coding is just an intermediate step in creating a bag of codewords representation for the whole image. Thus, sparse coding of each visual descriptor does not guarantee sparse coding of the whole image. This might prevent the use of such methods in real large scale applications that are constrained by either time or space resources. In this study, we propose and evaluate the mixed-norm regularizers [12, 10, 2] to take into account the structure of bags of visual descriptors present in images. Using this approach, we can for example specify an encoder that exploits the fact that once a codeword has been selected to help represent one of the visual descriptors of an image, it may as well be used to represent other visual descriptors of the same image without much additional regularization cost.

Furthermore, while images are represented as bags, the same idea could be used for *sets of images*, such as all the images from a given category. In this case, mixed regularization can be used to specify that when a codeword has been selected to help represent one of the visual descriptors of an image of a given category, it could as well be used to represent other visual descriptors of any image of the same category at no additional regularization cost. This form of regularization thus promotes the use of a small subset of codewords for each category that could be different from category to category, thus including an indirect discriminative signal in code construction.

Mixed regularization can be applied at two levels: for image encoding, which can be expressed as a convex optimization problem, and for dictionary learning, using an alternating minimization procedure. Dictionary regularization promotes a small dictionary size directly, instead of indirectly through the sparse encoding step.

The paper is organized as follows: Sec. 2 introduces the notation used in the rest of the paper, and summarizes the technical approach. Sec. 3 describes and solves the convex optimization problem for mixed-regularization encoding. Sec. 4 extends the technique to learn the dictionary by alternating optimization. Finally, Sec. 5 presents experimental results on a well-known image database.

## 2   Problem Statement

We denote scalars with lower-case letters, vectors with bold lower-case letters such as $\boldsymbol{v}$. We assume that the instance space is $\mathbb{R}^n$ endowed with the standard inner product between two vectors $\boldsymbol{u}$ and $\boldsymbol{v}$, $\boldsymbol{u} \cdot \boldsymbol{v} = \sum_{j=1}^{n} u_j v_j$. We also use the standard $\ell_p$ norms $\| \cdot \|_p$ over $\mathbb{R}^n$ with $p \in 1, 2, \infty$. We often make use of the fact that $\boldsymbol{u} \cdot \boldsymbol{u} = \|\boldsymbol{u}\|^2$, where as usual we omit the norm subscript for $p = 2$..

Our main goal is to encode effectively groups of instances in terms of a set of dictionary codewords $\mathcal{D} = \{\boldsymbol{d}_j\}_{j=1}^{|\mathcal{D}|}$. For example, if instances are image patches, each group may be the set of patches in a particular image, and each codeword may represent some kind of average patch. The $m$'th group is denoted $\mathcal{G}_m$ where $\mathcal{G}_m = \{\boldsymbol{x}_{m,i}\}_{i=1}^{|\mathcal{G}_m|}$ where each $\boldsymbol{x}_{m,i} \in \mathbb{R}^n$ is an instance. When discussing operations on a single group, we use $\mathcal{G}$ for the group in discussion and denote by $\boldsymbol{x}_i$ its $i$'th instance.

Given $\mathcal{D}$ and $\mathcal{G}$, our first subgoal, encoding, is to minimize a tradeoff between the reconstruction error for $\mathcal{G}$ in terms of $\mathcal{D}$, and a suitable mixed norm for the matrix of reconstruction weights that express each $\boldsymbol{x}_i$ as a positive linear combination of $\boldsymbol{d}_j \in \mathcal{D}$. The tradeoff between accurate reconstruction or compact encoding is governed through a regularization parameter $\lambda$.

Our second subgoal, learning, is to estimate a good dictionary $\mathcal{D}$ given a set of training groups $\{\mathcal{G}_m\}_{m=1}^{n}$. We achieve these goals by alternating between (i) fixing the dictionary to find recon-

struction weights that minimize the sum of encoding objectives for all groups, and (ii) fixing the reconstruction weights for all groups to find the dictionary that minimizes a tradeoff between the sum of group encoding objectives and the mixed norm of the dictionary.

## 3 Group Coding

To encode jointly all the instances in a group $\mathcal{G}$ with dictionary $\mathcal{D}$, we solve the following convex optimization problem:

$$
\begin{aligned}
\mathcal{A}^\star = & \arg\min_{\mathcal{A}} Q(\mathcal{A}, \mathcal{G}, \mathcal{D}) \\
\text{where} \quad Q(\mathcal{A}, \mathcal{G}, \mathcal{D}) &= \tfrac{1}{2} \sum_{i \in \mathcal{G}} \left\| \boldsymbol{x}_i - \sum_{j=1}^{|\mathcal{D}|} \alpha_j^i \boldsymbol{d}_j \right\|^2 + \lambda \sum_{j=1}^{|\mathcal{D}|} \|\boldsymbol{\alpha}_j\|_p \\
\text{and} \quad \alpha_j^i &\geq 0 \quad \forall i, j \ .
\end{aligned}
\tag{1}
$$

The reconstruction matrix $\mathcal{A} = \{\boldsymbol{\alpha}_j\}_{j=1}^{|\mathcal{D}|}$ consists of non-negative vectors $\boldsymbol{\alpha}_j = (\alpha_j^1, \ldots, \alpha_j^{|\mathcal{G}|})$ specifying the contribution of $\boldsymbol{d}_j$ to each instance. The second term of the objective weighs the mixed $\ell_1/\ell_p$ norm of $\mathcal{A}$, which measures reconstruction complexity, with the regularization parameter $\lambda$ that balances reconstruction quality (the first term) and reconstruction complexity.

The problem of Eq. (1) can be solved by coordinate descent. Leaving all indices intact except for index $r$, omitting fixed arguments of the objective, and denoting by $c_1$ and $c_2$ terms which do not depend on $\boldsymbol{\alpha}_r$, we obtain the following reduced objective:

$$
\begin{aligned}
Q(\boldsymbol{\alpha}_r) =& \frac{1}{2} \sum_{i \in \mathcal{G}} \left\| \boldsymbol{x}_i - \sum_{j \neq r} \alpha_j^i \boldsymbol{d}_j - \alpha_r^i \boldsymbol{d}_r \right\|^2 + \lambda \|\boldsymbol{\alpha}_r\|_p + c_1 \\
=& \sum_{i \in \mathcal{G}} \left( \sum_{j \neq r} \alpha_j^i \alpha_r^i (\boldsymbol{d}_j \cdot \boldsymbol{d}_r) - \alpha_r^i (\boldsymbol{x}_i \cdot \boldsymbol{d}_r) + \frac{1}{2}(\alpha_r^i)^2 \|\boldsymbol{d}_r\|^2 \right) + \lambda \|\boldsymbol{\alpha}_r\|_p + c_2 \ .
\end{aligned}
\tag{2}
$$

We next show how to find the optimum $\boldsymbol{\alpha}_r$ for $p = 1$ and $p = 2$. Let $\tilde{Q}$ be just the reconstruction term of the objective. Its partial derivatives with respect to each $\alpha_r^i$ are

$$
\frac{\partial}{\partial \alpha_r^i} \tilde{Q} = \sum_{j \neq r} \alpha_j^i (\boldsymbol{d}_j \cdot \boldsymbol{d}_r) - \boldsymbol{x}_i \cdot \boldsymbol{d}_r + \alpha_r^i \|\boldsymbol{d}_r\|^2 \ .
\tag{3}
$$

Let us make the following abbreviation for a given index $r$,

$$
\mu_i = \boldsymbol{x}_i \cdot \boldsymbol{d}_r - \sum_{j \neq r} \alpha_j^i (\boldsymbol{d}_j \cdot \boldsymbol{d}_r) \ .
\tag{4}
$$

It is clear that if $\mu_i \leq 0$ then the optimum for $\alpha_r^i$ is zero. In the derivation below we therefore employ $\mu_i^+ = [\mu_i]_+$ where $[z]_+ = \max\{0, z\}$. Next we derive the optimal solution for each of the norms we consider starting with $p = 1$. For $p = 1$ the objective function is separable and we get the following sub-gradient condition for optimality,

$$
0 \in -\mu_i^+ + \alpha_r^i \|\boldsymbol{d}_r\|^2 + \lambda \underbrace{\frac{\partial}{\partial \alpha_r^i} |\alpha_r^i|}_{\in [0,1]} \quad \Rightarrow \quad \alpha_r^i \in \frac{\mu_i^+ - [0, \lambda]}{\|\boldsymbol{d}_r\|^2} \ .
\tag{5}
$$

Since $\alpha_i^r \geq 0$ the above subgradient condition for optimality implies that $\alpha_i^r = 0$ when $\mu_i^+ \leq \lambda$ and otherwise $\alpha_i^r = (\mu_i^+ - \lambda)/\|\boldsymbol{d}_r\|^2$.

The objective function is not separable when $p = 2$. In this case we need to examine the entire set of values $\{\mu_i^+\}$. We denote by $\boldsymbol{\mu}^+$ the vector whose $i$'th value is $\mu_i^+$. Assume for now that the optimal solution has a non-zero norm, $\|\boldsymbol{\alpha}_r\|_2 > 0$. In this case, the gradient of $Q(\boldsymbol{\alpha}_r)$ with an $\ell_2$ regularization term is

$$
\|\boldsymbol{d}_r\|^2 \boldsymbol{\alpha}_r - \boldsymbol{\mu}^+ + \lambda \frac{\boldsymbol{\alpha}_r}{\|\boldsymbol{\alpha}_r\|} \ .
$$

At the optimum this vector must be zero, so after rearranging terms we obtain

$$\boldsymbol{\alpha}_r = \left( \|\boldsymbol{d}_r\|^2 + \frac{\lambda}{\|\boldsymbol{\alpha}_r\|} \right)^{-1} \boldsymbol{\mu}^+ \ . \tag{6}$$

Therefore, the vector $\boldsymbol{\alpha}_r$ is in the same direction as $\boldsymbol{\mu}^+$ which means that we can simply write $\boldsymbol{\alpha}_r = s\,\boldsymbol{\mu}^+$ where $s$ is a non-negative scalar. We thus can rewrite Eq. (6) solely as a function of the scaling parameter $s$

$$s\,\boldsymbol{\mu}^+ = \left( \|\boldsymbol{d}_r\|^2 + \frac{\lambda}{s\|\boldsymbol{\mu}^+\|} \right)^{-1} \boldsymbol{\mu}^+ \ ,$$

which implies that

$$s = \frac{1}{\|\boldsymbol{d}_r\|^2} \left( 1 - \frac{\lambda}{\|\boldsymbol{\mu}^+\|} \right) \ . \tag{7}$$

We now revisit the assumption that the norm of the optimal solution is greater than zero. Since $s$ cannot be negative the above expression also provides the condition for obtaining a zero vector for $\boldsymbol{\alpha}_r$. Namely, the term $1 - \lambda/\|\boldsymbol{\mu}^+\|$ must be positive, thus, we get that $\boldsymbol{\alpha}_r = \boldsymbol{0}$ if $\|\boldsymbol{\mu}^+\| \leq \lambda$ and otherwise $\boldsymbol{\alpha}_r = s\boldsymbol{\mu}^+$ where $s$ is defined in Eq. (7).

Once the optimal group reconstruction matrix $\mathcal{A}$ is found, we compress the matrix into a single vector. This vector is of fixed dimension and does not depend on the number of instances that constitute the group. To do so we simply take the $p$-norm of each $\boldsymbol{\alpha}_j$, thus yielding a $|\mathcal{D}|$ dimensional vector. Since we use mixed-norms which are sparsity promoting, in particular the $\ell_1/\ell_2$ mixed-norm, the resulting vector is likely to be sparse, as we show experimentally in Section 6.

Since visual descriptors and dictionary elements are only accessed through inner products in the above method, it could be easily generalized to work with Mercer kernels instead.

## 4  Dictionary Learning

Now that we know how to achieve optimal reconstruction for a given dictionary, we examine how to learn a good dictionary, that is, a dictionary that balances between reconstruction error, reconstruction complexity, overall complexity relative to the given training set. In particular, we seek a learning method that facilitates both induction of new dictionary words and the removal of dictionary words with low predictive power. To achieve this goal, we will apply $\ell_1/\ell_2$ regularization controlled by a new hyperparameter $\gamma$, to dictionary words. For this approach to work, we assume that instances have been mean-subtracted so that the zero vector $\boldsymbol{0}$ is the (uninformative) mean of the data and regularization towards $\boldsymbol{0}$ is equivalent to removing words that do not contribute much to compact representation of groups.

Let $\boldsymbol{G} = \{\mathcal{G}_1, \ldots, \mathcal{G}_n\}$ be a set of groups and $\boldsymbol{A} = \{\mathcal{A}_1, \ldots, \mathcal{A}_n\}$ the corresponding reconstruction coefficients relative to dictionary $\mathcal{D}$. Then, the following objective meets the above requirements:

$$Q(\boldsymbol{A}, \mathcal{D}) = \sum_{m=1}^{n} Q(\mathcal{A}_m, \mathcal{G}_m, \mathcal{D}) + \gamma \sum_{k=1}^{|\mathcal{D}|} \|\boldsymbol{d}_k\|_p \ \text{s.t.}\ \alpha_{m,j}^i \geq 0\ \forall i, j, m \ , \tag{8}$$

where the single group objective $Q(\mathcal{A}_m, \mathcal{G}_m, \mathcal{D})$ is as in Eq. (1).

In our application we set $p = 2$ as the norm penalty of the dictionary words. For fixed $\boldsymbol{A}$, the objective above is convex in $\mathcal{D}$. Moreover, the same coordinate descent technique described above for finding the optimum reconstruction weights can be used again here after simple algebraic manipulations. Define the following auxiliary variables:

$$\boldsymbol{v}_r = \sum_m \sum_i \alpha_{m,r}^i \boldsymbol{x}_{m,i} \ \text{and}\ \nu_{j,k} = \sum_m \sum_i \alpha_{m,j}^i \alpha_{m,k}^i \ . \tag{9}$$

Then, we can express $\boldsymbol{d}_r$ compactly as follows. As before, assume that $\|\boldsymbol{d}_r\| > 0$. Calculating the gradient with respect to each $\boldsymbol{d}_r$ and equating it to zero, we obtain

$$\sum_m \sum_{i \in \mathcal{G}_m} \left( \sum_{j \neq r} \alpha_{m,j}^i \alpha_{m,r}^i \boldsymbol{d}_j + (\alpha_{m,r}^i)^2 \boldsymbol{d}_r - \alpha_{m,r}^i \boldsymbol{x}_{m,i} \right) + \gamma \frac{\boldsymbol{d}_r}{\|\boldsymbol{d}_r\|} = 0 \ .$$

Swapping the sums over $m$ and $i$ with the sum over $j$, using the auxiliary variables, and noting that $\boldsymbol{d}_j$ does not depend neither on $m$ nor on $i$, we obtain

$$\sum_{j \neq r} \nu_{j,r} \boldsymbol{d}_j + \nu_{r,r} \boldsymbol{d}_r - \boldsymbol{v}_r + \gamma \frac{\boldsymbol{d}_r}{\|\boldsymbol{d}_r\|} = \boldsymbol{0} \ . \tag{10}$$

Similarly to the way we solved for $\boldsymbol{\alpha}_r$, we now define the vector $\boldsymbol{u}_r = \boldsymbol{v}_r - \sum_{j \neq r} \nu_{j,r} \boldsymbol{d}_j$ to get the following iterate for $\boldsymbol{d}_r$:

$$\boldsymbol{d}_r = \nu_{r,r}^{-1} \left[ 1 - \frac{\gamma}{\|\boldsymbol{u}_r\|} \right]_+ \boldsymbol{u}_r \ , \tag{11}$$

where, as above, we incorporated the case $\boldsymbol{d}_r = \boldsymbol{0}$, by applying the operator $[\cdot]_+$ to the term $1 - \gamma/\|\boldsymbol{u}_r\|$. The form of the solution implies that we can eliminate $\boldsymbol{d}_r$, as it becomes $\boldsymbol{0}$, whenever the norm of the residual vector $\boldsymbol{u}_r$ is smaller than $\gamma$. Thus, the dictionary learning procedure naturally facilitates the ability to remove dictionary words whose predictive power falls below the regularization parameter.

## 5 Experimental Setting

We compare our approach to image coding with previous sparse coding methods by measuring their impact on classification performance on the PASCAL VOC (Visual Object Classes) 2007 dataset [4]. The VOC datasets contain images from 20 classes, including people, animals (*bird*), vehicles (*aeroplane*), and indoor objects (*chair*), and are considered natural, difficult images for classification. There are around 2500 training images, 2500 validation images and 5000 test images in total.

For each coding technique under consideration, we explore a range of values for the hyperparameters $\lambda$ and $\gamma$. In the past, many features have been used for VOC classification, with bag-of-words histograms of local descriptors like SIFT [6] being most popular. In our experiments, we extract local descriptors based on a regular grid for each image. The grid points are located at every seventh pixel horizontally and vertically, which produces an average of 3234 descriptors per image. We used a custom local descriptor that collects Gabor wavelet responses at different orientations, spatial scales, and spatial offsets from the interest point. Four orientations ($0°$, $45°$, $90°$, $135°$) and 27 (scale, offset) combinations are used, for a total of 108 components. The 27 (scale, offset) pairs were chosen by optimizing a previous image recognition task, unrelated to this paper, using a genetic algorithm. Tola *et al.* [15] independently described a descriptor that similarly uses responses at different orientations, scales, and offsets (see their Figure 2). Overall, this descriptor is generally comparable to SIFT and results in similar performance.

To build an image feature vector from the descriptors, we thus investigate the following methods:

1. Build a bag-of-words histogram over hierarchical $k$-means codewords by looking up each descriptor in a hierarchical $k$-means tree [11]. We use branching factors of 6 to 13 and a depth of 3 for a total of between 216 and 2197 codewords. When used with multiple feature types, this method results in very good classification performance on the VOC task.

2. Jointly train a dictionary and encode each descriptor using an $\ell_1$ sparse coding approach with $\gamma = 0$, which was studied previously [5, 7, 9].

3. Jointly train a dictionary and encode sets of descriptors where each set corresponds to a single image, using $\ell_1/\ell_2$ group sparse coding, varying both $\gamma$ and $\lambda$.

4. Jointly train a dictionary and encode sets of descriptors where each set corresponds to all descriptors or all images of a single class, using $\ell_1/\ell_2$ sparse coding, varying both $\gamma$ and $\lambda$. Then, use $\ell_1/\ell_2$ sparse coding to encode the descriptors in individual images and obtain a single $\boldsymbol{\alpha}$ vector per image.

As explained before, we normalized all descriptors to have zero mean so that regularizing dictionary words towards the zero vector implies dictionary sparsity.

In all cases, the initial dictionary used during training was obtained from the same hierarchical $k$-means tree, with a branching factor of 10 and depth 4 rather than 3 as used in the baseline method. This scheme yielded an initial dictionary of size 7873.

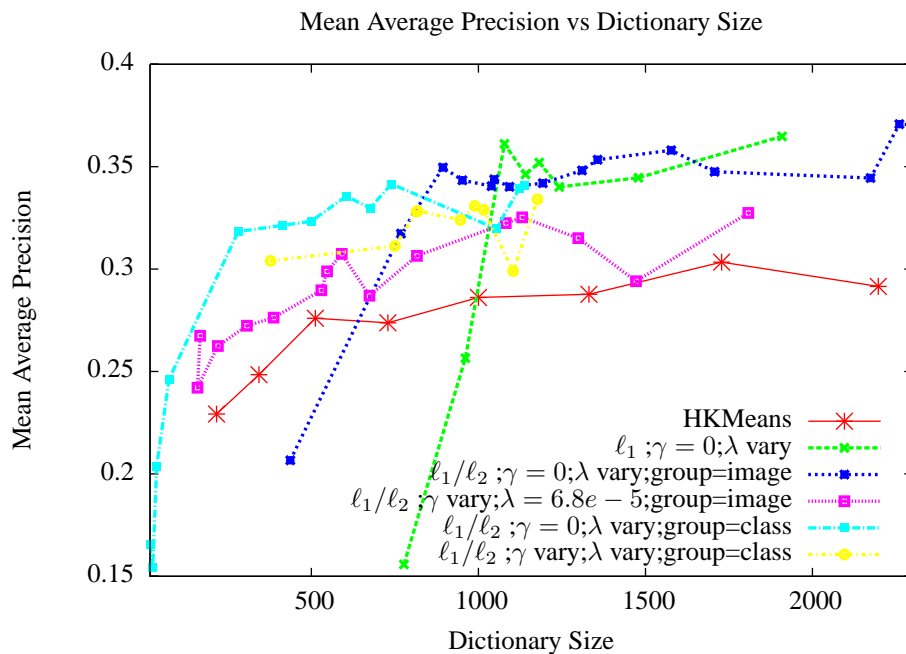

Figure 1: Mean Average Precision on the 2007 PASCAL VOC database as a function of the size of the dictionary obtained by both $\ell_1$ and $\ell_1/\ell_2$ regularization approaches when varying $\lambda$ or $\gamma$. We show results where descriptors are grouped either by image or by class. The baseline system using hierarchical $k$-means is also shown.

To evaluate the impact of different coding methods on an important end-to-end task, image classification, we selected the VOC 2007 training set for classifier training, the VOC 2007 validation set for hyperparameter selection, and the VOC 2007 test set for for evaluation. After the datasets are encoded with each of the methods being evaluated, a one-versus-all linear SVM is trained on the encoded training set for each of the 20 classes, and the best SVM hyperparameter $C$ is chosen on the validation set. Class average precisions on the encoded test set are then averaged across the 20 classes to produce the mean average precision shown in our graphs.

## 6 Results and Discussion

In Figure 1 we compare the mean average precisions of the competing approaches as encoding hyperparameters are varied to control the overall dictionary size. For the $\ell_1$ approach, achieving different dictionary size was obtained by tuning $\lambda$ while setting $\gamma = 0$. For the $\ell_1/\ell_2$ approach, since it was not possible to compare all possible combinations of $\lambda$ and $\gamma$, we first fixed $\gamma$ to be zero, so that it could be comparable to the standard $\ell_1$ approach with the same setting. Then we fixed $\lambda$ to a value which proved to yield good results and varied $\gamma$. As it can be seen in Figure 1, when the dictionary is allowed to be very large, the pure $\ell_1$ approach yields the best performance. On the other hand, when the size of the dictionary matters, then all the approaches based on $\ell_1/\ell_2$ regularization performed better than the $\ell_1$ counterpart. Even hierarchical $k$-means performed better than the pure $\ell_1$ in that case. The version of $\ell_1/\ell_2$ in which we allowed $\gamma$ to vary provided the best tradeoff between dictionary size and classification performance when descriptors were grouped per image, which was to be expected as $\gamma$ directly promotes sparse dictionaries. More interestingly, when grouping descriptors per class instead of per image, we get even better performance for small dictionary sizes by varying $\lambda$.

In Figure 2 we compare the mean average precisions of $\ell_1$ and $\ell_1/\ell_2$ regularization as average image size varies. When image size is constrained, which is often the case is large-scale applications, all

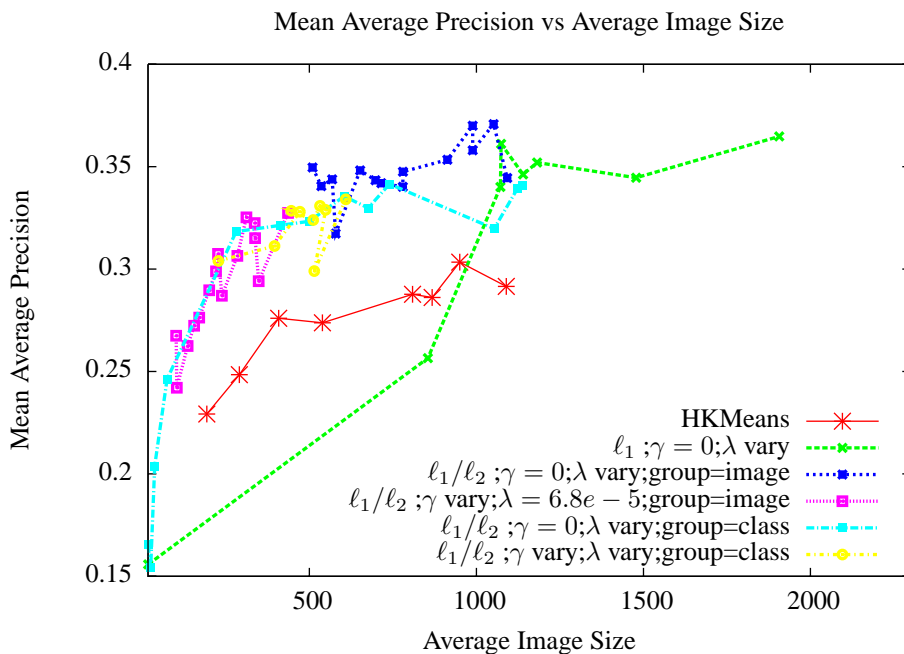

Figure 2: Mean Average Precision on the 2007 PASCAL VOC database as a function of the average size of each image as encoded using the trained dictionary obtained by both $\ell_1$ and $\ell_1/\ell_2$ regularization approaches when varying $\lambda$ and $\gamma$. We show results where descriptors are grouped either by image or by class. The baseline system using hierarchical $k$-means is also shown.

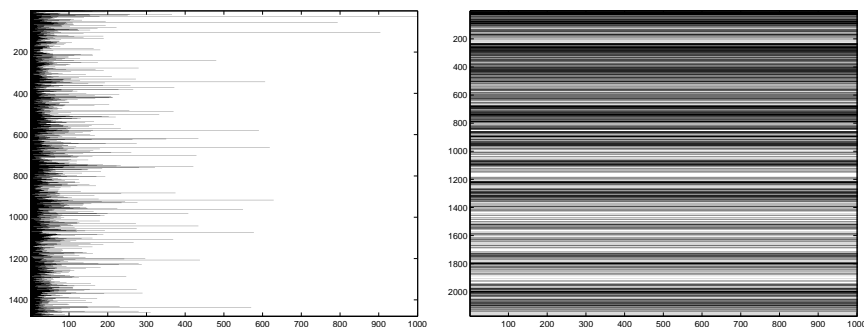

Figure 3: Comparison of the dictionary words used to reconstruct the same image. A pure $\ell_1$ coding was used on the left, while a mixed $\ell_1/\ell_2$ encoding was used on the right plot. Each row represents the number of times each dictionary word was used in the reconstruction of the image.

the $\ell_1/\ell_2$ regularization choices yield better performance than $\ell_1$ regularization. Once again $\ell_1$ regularization performed even worse than hierarchical $k$-means for small image sizes

Figure 3 compares the usage of dictionary words to encode the same image, either using $\ell_1$ (on the left) or $\ell_1/\ell_2$ (on the right) regularization. Each graph shows the number of times a dictionary word (a row in the plot) was used in the reconstruction of the image. Clearly, $\ell_1$ regularization yields an overall sparser representation in terms of total number of dictionary coefficients that are used. However, almost all of the resulting dictionary vectors are non-zero and used at least once in the coding process. As expected, with $\ell_1/\ell_2$ regularization, a dictionary word is either always used or never used yielding a much more compact representation in terms of the total number of dictionary words that are used.

Overall, mixed-norm regularization yields better performance when the problem to solve includes resource constraints, either time (a smaller dictionary yields faster image encoding) or space (one can store or convey more images when they take less space). They might thus be a good fit when a tradeoff between pure performance and resources is needed, as is often the case for large-scale applications or online settings.

Finally, grouping descriptors per class instead of per image during dictionary learning promotes the use of the same dictionary words for all images of the same class, hence yielding some form of weak discrimination which appears to help under space or time constraints.

## Acknowledgments

We would like to thanks John Duchi for numerous discussions and suggestions.

## References

[1] R. Baeza-Yates and B. Ribeiro-Neto. *Modern Information Retrieval*. Addison Wesley, Harlow, England, 1999.

[2] J. Duchi and Y. Singer. Boosting with structural sparsity. In *International Conference on Machine Learning (ICML)*, 2009.

[3] M. Elad and M. Aharon. Image denoising via sparse and redundant representation over learned dictionaries. *IEEE Transaction on Image Processing*, 15(12):3736–3745, 2006.

[4] M. Everingham, L. Van Gool, C. K. I. Williams, J. Winn, and A. Zisserman. The PASCAL Visual Object Classes Challenge 2007 (VOC2007) Results. http://www.pascal-network.org/challenges/VOC/voc2007/workshop/index.html.

[5] H. Lee, A. Battle, R. Raina, and A. Y. Ng. Efficient sparse coding algorithms. In *Advances in Neural Information Processing Systems (NIPS)*, 2007.

[6] D. G. Lowe. Object recognition from local scale-invariant features. In *International Conference on Computer Vision (ICCV)*, pages 1150–1157, 1999.

[7] J. Mairal, F. Bach, J. Ponce, and G. Sapiro. Online dictionary learning for sparse coding. In *International Conference on Machine Learning (ICML)*, 2009.

[8] J. Mairal, M. Elad, and G. Sapiro. Sparse representation for color image restoration. *IEEE Transaction on Image Processing*, 17(1), 2008.

[9] J. Mairal, M. Leordeanu, F. Bach, M. Hebert, and J. Ponce. Discriminative sparse image models for class-specific edge detection and image interpretation. In *European Conference on Computer Vision (ECCV)*, 2008.

[10] S. Negahban and M. Wainwright. Phase transitions for high-dimensional joint support recovery. In *Advances in Neural Information Processing Systems 22*, 2008.

[11] D. Nister and H. Stewenius. Scalable recognition with a vocabulary tree. In *Proceedings of the IEEE Conference on Computer Vision and Pattern Recognition (CVPR)*, 2006.

[12] G. Obozinski, B. Taskar, and M. Jordan. Joint covariate selection for grouped classification. Technical Report 743, Dept. of Statistics, University of California Berkeley, 2007.

[13] B. A. Olshausen and D. J. Field. Sparse coding with an overcomplete basis set: A strategy employed by v1? *Vision Research*, 37, 1997.

[14] P. Quelhas, F. Monay, J. M. Odobez, D. Gatica-Perez, T. Tuytelaars, and L. J. Van Gool. Modeling scenes with local descriptors and latent aspects. In *International Conference on Computer Vision (ICCV)*, 2005.

[15] E. Tola, V. Lepetit, and P. Fua. A fast local descriptor for dense matching. In *Proceedings of the IEEE Conference on Computer Vision and Pattern Recognition (CVPR)*, 2008.

[16] J. yang, K. Yu, Y. Gong, and T. Huang. Linear spatial pyramid matching using sparse coding for image classification. In *Proceedings of the IEEE Conference on Computer Vision and Pattern Recognition (CVPR)*, 2009.

